# Hallucinations in Charles Bonnet Syndrome Induced by Homeostasis: a Deep Boltzmann Machine Model

**David P. Reichert, Peggy Series and Amos J. Storkey**
School of Informatics, University of Edinburgh
10 Crichton Street, Edinburgh, EH8 9AB
{d.p.reichert@sms., pseries@inf., a.storkey@} ed.ac.uk

## Abstract

The Charles Bonnet Syndrome (CBS) is characterized by complex vivid visual hallucinations in people with, primarily, eye diseases and no other neurological pathology. We present a Deep Boltzmann Machine model of CBS, exploring two core hypotheses: First, that the visual cortex learns a generative or predictive model of sensory input, thus explaining its capability to generate internal imagery. And second, that homeostatic mechanisms stabilize neuronal activity levels, leading to hallucinations being formed when input is lacking. We reproduce a variety of qualitative findings in CBS. We also introduce a modification to the DBM that allows us to model a possible role of acetylcholine in CBS as mediating the balance of feed-forward and feed-back processing. Our model might provide new insights into CBS and also demonstrates that generative frameworks are promising as hypothetical models of cortical learning and perception.

## 1 Introduction

Complex visual hallucinations [1] can offer a fascinating insight into how the brain realizes visual perception. The content of such hallucinations can be highly elaborate, consisting of people, animals, objects and whole scenes, and the images supposedly can "exceed anything seen in real life" in detail and vividness [1]. Attempts have been made to unify complex hallucinations in various pathologies in one qualitative model, but many argue that the underlying causal mechanisms are too varied to do so [2]. Of particular interest is the Charles Bonnet Syndrome (CBS) [3, 4, 5], where patients experience complex visual hallucinations which appear not to be causally related to any other impairment to mental health and where the primary pathology is one of loss of vision due to eye diseases. Sensory deprivation is thus implicated as playing a key role in the development of CBS, and comparisons have been made to phantom limb phenomena [3, 5].

The mechanisms behind complex hallucinations remain obscure. Theories of CBS are descriptive in nature and no computational model exists. For example, hallucinations are attributed to 'perceptual traces being released' [3] that would normally be inhibited by sensory input, or it is suggested that experience in general is evoked by internally generated neuronal activity in distributed networks, a 'neuromatrix', imposing meaning on sensory input or onto unspecific input in the case of hallucinations [3, 5]. The phenomenon of internally generated images becomes less mysterious if one assumes the cortex implements an actual generative model of sensory input. The hypothesis that cortical learning is driven by prediction or reconstruction of sensory input is promising as it could explain how the brain might learn in an unsupervised fashion, evaluating its internal model of the world by matching predictions to actual input [6, 7]. Consequently, the idea that disorders including hallucinations are caused by mismatches between internally generated expectations and sensory input has recently found interest in psychology [8, 9]. If generating internal imagery is an essential aspect of normal vision, then this could explain why hallucinations occur in so many different pathologies, even sometimes when there is no direct malfunction of the visual system itself [1].

One modeling framework that implements unsupervised generative learning in a neural architecture is the Deep Boltzmann Machine (DBM) [10]. DBMs have been developed in a machine learning context, but we argue that they could model aspects of cortical learning and perception as well. They are related to Hopfield networks, which have been used in the context of models of hallucinations before [11]. However, whereas the latter model some abstract memory system, DBMs (and the related Deep Belief Nets) learn hierarchical representations of data [7], thus capturing aspects of the visual cortex [12], the locus where visual hallucinations are ultimately realized [1, 13]. We aim to relate inference in a DBM to mechanisms of cortical perception.

We thus present a DBM model of the CBS, and propose a concrete mechanism that could lead to hallucinations being formed: homeostasis. There is strong experimental evidence that homeostatic processes serve to stabilize the activity level of neuronal populations through a variety of cellular and synaptic mechanisms [14]. Moreover, deafferentiated cortex becomes hyper-excitable, and it has been suggested before that this could be a result of homeostasis [15]. Hence, in CBS a lack of visual input could lead to intrinsic excitability changes of neurons setting in to restore original activity levels. In our model, we demonstrate how these changes could cause spontaneous 'complex' hallucinations to be formed even when sensory input is lacking. These hallucinations are complex in the sense that they involve learned, distributed representations of objects in (toy) images rather than, for example, corresponding to local structural features of topographically organized cortical areas, the latter being implicated in simpler hallucinations such as geometric patterns [16].

In Section 3, we first show that homeostasis can be beneficial in a DBM, enabling the model to recover correct internal representations from degraded input. Then we move on to hallucinations. The CBS is a complex phenomenon and differs considerably among patients, but we can qualitatively reproduce several aspects found in most or some cases: An initial latent period after loss of vision that is free of hallucinations; a localization of hallucinations to lesioned parts of the visual field (Section 3.1), potentially also explaining a tendency to see hallucinated objects too small for their surroundings; and, effects of cortical lesions and cortical suppression of activity (Section 3.2). Moreover, hallucinations in CBS tend to occur more often in states of drowsiness, implicating a role of cholinergic and serotonergic systems [1]. By introducing a modification to the DBM model, we can account for this by taking acetylcholine to modulate the bottom-up top-down balance of information flow (Section 4). Finally, we speculate on the potential of the DBM to model other pathologies and on the difference between hallucinations and mental imagery (Section 5).

## 2 Deep Boltzmann Machines

Boltzmann machines (BMs) [17] are closely related to Hopfield networks, which have been employed as models of hallucinations before (e.g. [18]). Both models consist of neural units $\mathbf{x}$ (here binary with value 1 or 0, on or off) connected with symmetric weights $\mathbf{W}$. A unit's state is determined by a sigmoid activation function, and biases $\mathbf{b}$ control the excitability of each unit. The overall state of the network evolves according to an 'energy' function, $E(\mathbf{x}) = -\mathbf{x}^T \mathbf{W} \mathbf{x} - \mathbf{b}^T \mathbf{x}$, where minima in the energy landscape correspond to attractor states. A BM differs from a Hopfield net in two important points. First, a BM is stochastic in that the activation of a unit determines the probability for it to switch on:

$$P(x_i = 1|\mathbf{x}) = \sigma(\sum_j w_{ij} x_j + b_i) = \frac{1}{1 + \exp(-\sum_j w_{ij} x_j - b_i)}. \tag{1}$$

When the model is run by switching units on and off stochastically, it performs a random walk in the energy landscape. Asymptotically, any state $\mathbf{x}$ will be assumed with probability $P(\mathbf{x}) \propto \exp(-E(\mathbf{x}))$. Hence, a BM can be understood as modeling the probability distribution of the data rather than just as a memory network, which is why these models are of interest in machine learning.

The second difference is the possible introduction of *hidden* units, separating $\mathbf{x}$ into visible units $\mathbf{v}$ and hidden units $\mathbf{h}$. Whereas the former represent visible variables such as the pixels of an image, the latter represent latent variables that help to explain the image. Learning in a BM is then performed with the aim of forming hidden representations from which data can be generated/predicted/reconstructed. In modeling $P(\mathbf{x})$ for any $\mathbf{x}$, not just data seen in training, one goal is to learn latent representations that make it possible to generalize over novel data. Another goal is to learn representations which can then be utilized further, for example for classification or clustering

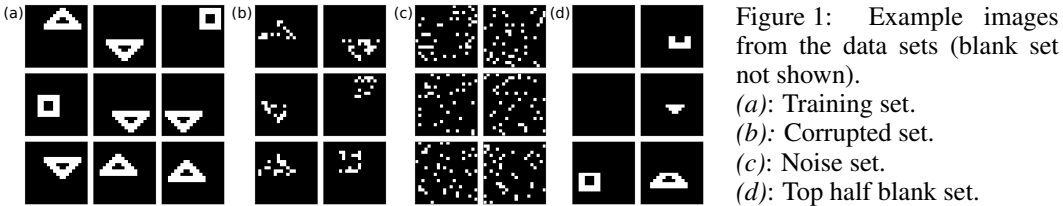

Figure 1: Example images from the data sets (blank set not shown).
*(a)*: Training set.
*(b):* Corrupted set.
*(c)*: Noise set.
*(d)*: Top half blank set.

[19]. The modeling context of a BM is thus rather different from that of a Hopfield network.

A Deep Boltzmann Machine (DBM) [10] is a BM with a special architecture (Figure 2a) consisting of a visible layer and several subsequent hidden layers stacked on top of each other. To simplify computations there are no lateral connections within any layer. When trained on a data set of images, each pair of adjacent layers is trained one at a time so that each layer learns to generate the activations of the layer below, using only biologically plausible local Hebbian (and anti-Hebbian) weight changes. See [10] for details.[1] Furthermore, to make a more concrete connection to the visual cortex, we impose a hierarchical receptive field structure on the model: Each layer's units are arranged topographically, and each unit's weights are restricted so that it receives inputs only from a square patch of units below. In detail, the model had 20x20 visible units corresponding to images with 20x20 pixels, and three hidden layers of 26x26 units each. Each unit in the first hidden layer received inputs from a 7x7 patch of visible units, whereas the higher layers received inputs from half (13x13) and all (26x26) of the units of the respective lower layer. The training data set used consisted of toy images (Figure 1a), containing individual shapes out of three categories (upwards triangles, downward triangles, squares) at random positions.

## 2.1 Sampling and decoding the internal state

To model perception, we clamp the visible units to an image and sample the hidden units, starting from the first hidden layer and proceeding to the topmost, then going downwards, and repeating this cycle. For each layer, all units can be sampled in parallel. Processing across the cortical hierarchy is suggested to be cyclic as well [20].

We are interested in the representations formed internally, in the hidden layers of the DBM, when visual input is fixed or lacking in the case of CBS. To decode the states of the hidden layers, we define a top-down projection to obtain a reconstructed image. Given the states $\mathbf{x}_k$ of the hidden layer $k$ in question, the activations $\mathbf{a}_{k-1}$ of the layer below are computed taking only $\mathbf{x}_k$ into account, ignoring the states $\mathbf{x}_{k-2}$ further below.[2] This process is repeated down to the visibles, so that we obtain a reconstructed image which has been determined only from the states in layer $k$ (using activations instead of samples to obtain less noisy images). Note that we perform the top-down projection only to inspect the internal states. For the actual inference procedure, all intermediate layers are always sampled properly taking both adjacent layers into account.[3] In this work, hidden states are always initialized to zero for each image and evaluated after 40 sampling cycles.

To evaluate the quality of an internal representation when a shape image is presented to the model, we compute the maximum value of the normalized cross-correlation of the projected reconstruction with that image. In the case of hallucinations, internal representations are matched against all three template shapes, taking the one that matches best as being represented and the corresponding cross-correlation value as measure of the quality of the hallucination (Figure 2b).

## 2.2 Homeostasis in a DBM

Homeostatic mechanisms in the cortex are found to stabilize neuronal activity [14]. In our model, we assume that neurons have an individual preferred activation level that is attained as representations of inputs are learned. Hence, after the model has been trained we compute each unit's activation averaged over 40 sample cycles over all training images, taking this as the 'healthy' activation level

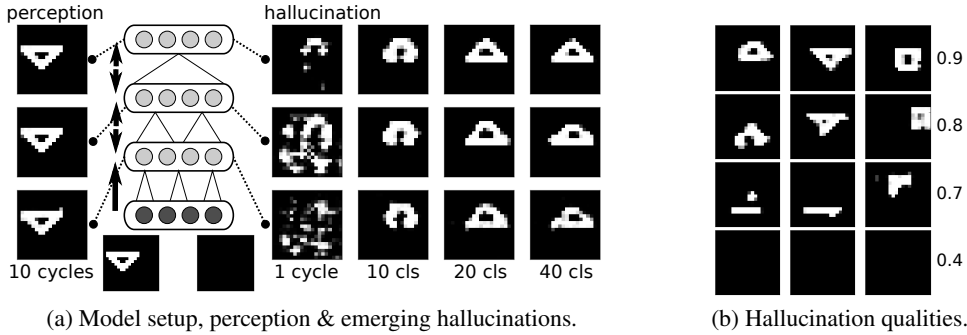

(a) Model setup, perception & emerging hallucinations.

(b) Hallucination qualities.

Figure 2: *(a) left-hand side*: Setup of the model and perception. With the visible units (dark) clamped to an image (bottom left), the hidden layer states assume representations of that image. Displayed are the decoded projections for each layer after ten recurrent cycles (left column). *(a) right-hand side*: After homeostatic regulation given empty images, hallucinations form spontaneously. Hallucinations are often stable after a few tens of recurrent cycles, but still fluctuate due to the stochastic nature of the DBM. They are slightly less well formed in lower layers, which require higher layer input to form stable shape perceptions. *(b)*: Examples of hallucinations of different qualities (computed from cross-correlations with templates). 1.0 is perfect match with template.

under normal sensory input. To simulate CBS, we then blank the visual input and let the model employ homeostatic regulation to recover healthy activation levels. The homeostatic mechanism is implemented in a straightforward way, adjusting only the biases (as in [12]) to model changes to intrinsic excitability of a neuron. Specifically, we present a mini-batch of 100 (corrupted or blank) images for 40 cycles each to compute each unit's new average activity $a_i$, and with the preferred activity $p_i$ modify the unit's bias $b_i$ according to[4]

$$\Delta b_i = \eta(p_i - a_i), \tag{2}$$

where $\eta$ is rate of change parameter (set to 0.1, but the precise value was not found to matter). This was repeated for 1000 iterations, and $\Delta b$ averaged over the population tended to zero before that as original activity levels were restored. We note that similar mechanisms have been employed in DBM-like models during training itself to enforce sparsity in the activations [12, 21], resulting in V1 and V2 like receptive fields being learned [12]. However such sparsity is not focus of this paper.

## 3 Hallucinations emerging due to homeostasis

First, we demonstrate that homeostasis can be a beneficial mechanism in a DBM. To this end, we present the trained model with heavily corrupted training images (Figure 1b) in which pixels where turned off with probability 0.65, emulating pre-cortical damage to the visual input. We then computed the reconstruction quality (Section 2.1) from the reconstructions projected from the top hidden layer states (after 40 cycles of sampling), and found it to be 0.46 on average. For comparison, average reconstruction quality on the uncorrupted training images was 0.98. The degradation of input was also reflected in changes in mean activities of the layers (Figure 3a). We then applied homeostasis as described in Section 2.2. As the excitability of the units was adjusted, mean activity levels for each hidden layer were gradually restored. At the same time, the reconstruction quality rose to about 0.9 (Figure 3a). Thus, a simple local activity stabilization can help alleviating damage to the system. Note that due to excitation and inhibition being mixed in the weights of a BM, a lack or degradation of input changes but not necessarily decreases activity. Homeostasis in a DBM thus restores activity levels in some cases by increasing and in some cases by attenuating neuronal excitability.

To model the CBS which is often triggered by profound retinal damage, we then repeated the homeostasis experiment with blank images. Now, any formation of internal representations could be regarded as hallucinations. The question was whether internal representations were stable and corresponded to 'objects' the model had learned, rather than random patterns. After all, the local changes of excitability and the permanent loss of visual input could have interfered with the dynamics of internal representations. As shown in Figure 3b, the activity levels of the hidden layers were restored

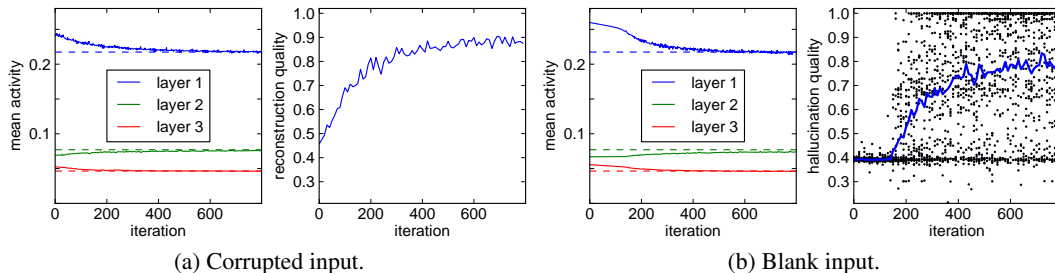

(a) Corrupted input.          (b) Blank input.

Figure 3: *(a)*: For corrupted images, homeostatic restoration of original activity levels (left figure, dashed lines) in the three hidden layers, and recovering reconstruction quality (right figure). *(b)*: For blank images, restoration of activity (left) and qualities of emerging hallucinations (right). Each dot marks one hallucination, plotted for 25 trials out of 100 per iteration. Blue curve is average quality.

by homeostasis. To analyze the internal representations of the model, we computed the qualities of individual representations of the topmost hidden layer from the projected reconstructions (Figure 3b). Over an initial period, internal representations correspond to blank images even though activity levels gradually improve. At some point however, hallucinations emerge, and relatively soon can they reach high quality levels. At this point activities start to change more rapidly, hence hallucinations themselves contribute to the restoration of activity levels. These results are consistent with CBS: If loss of vision is abrupt, hallucinations emerge after an initial latent period lasting hours to days [5], which matches well the time scale on which homeostatic mechanisms take place [14].

Besides a peak of hallucination quality at 1.0, there are also numerous hallucinations of lower quality, which could be in line with CBS as there complex hallucinations are often mixed with simple, less sophisticated ones. Also, some of the lower quality hallucinations are of transitory nature (if run for 200 instead of 40 cycles, mean quality rose from 0.83 to 0.88). Still, in Section 4 we will present mechanisms that lead to more stable hallucinations.

We also repeated the experiment with images containing random noise instead of being blank (Figure 1c) to simulate a different type of visual impairment than total blindness. We found in this case that smaller overall bias shifts were necessary to restore original activity levels (not shown) and produce hallucinations (Figure 5a). This shows that the exact nature of visual impairment could have an impact on whether and when hallucinations are formed. Indeed, many CBS patients develop hallucinations as vision degrades, but stop hallucinating when vision is finally lost completely [5]. In our model, this could be explained when one assumes there is a limit to how much neuronal excitability can be adapted. Thus, as long as there is some input, even if it is unspecific noise, hallucinations can be formed, but losing the input completely might require too much of a bias shift. Another reason for the cessation of hallucinations over time could be input specific synaptic plasticity, i.e. learning. If we were to train the model on empty images, it would learn to generate those. Hence, homeostasis as a short-term stabilization mechanism could lead to hallucinations, but a long-term reorganization of the cortex to represent the novel input would cause them to cease.

## 3.1 Localized hallucinations with localized lesions

Another property of the hallucinations was that the represented shapes were found to be distributed over the whole image and could be any of the three categories (Figure 4a). This is of relevance as complex hallucinations in CBS vary from episode to episode in a majority of patients [4]. Hence, it is important that the model can form a variety of internal representations of learned images instead of just converging to a few degenerate states.

Damage to the visual input leading to CBS can be restricted to parts of the visual field. Some studies [5] report that hallucinations tend to be localized to the blind regions. To test whether we could reproduce this, we repeated the homeostasis experiment with images from the training data set in which only the top half had been blanked out (Figure 1d), simulating a localized impairment of vision. As before, we found mean activities initially to be changed due to the degraded input and then be restored after homeostatic regulation. To test whether hallucinations would form at any location, we then tested the model on blank images. As shown in Figure 4b, the stable hallucinations were

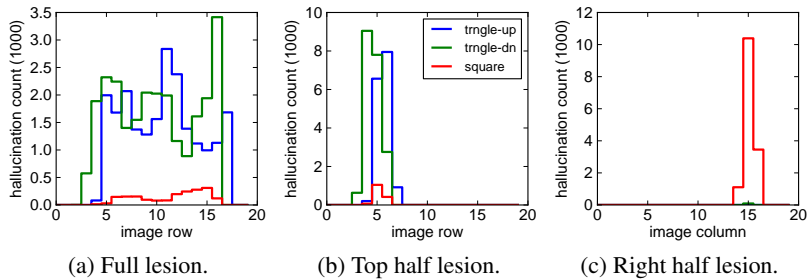

(a) Full lesion.      (b) Top half lesion.      (c) Right half lesion.

Figure 4: Localization (center of mass) of high quality (>0.95) hallucinations in projected images at the end of homeostatic regulation. Counts in thousands out of $10^5$ trials. *(a)*: Although local hotspots exist and squares are least likely to occur, overall hallucinations vary in type and location. *(b)*: When only the top half of visual input was blanked during homeostatic regulation, hallucinations emerged localized to the 'blind' half. *(c)*: When a region too narrow for triangles was blanked instead, hallucinations were almost always squares.

located in the top half of the visual field only, corresponding to the region of the 'lesion'. Excitability changes in the network are thus specific enough to have topographic properties.

This result lets us speculate on another phenomenon of CBS. In some cases, hallucinated objects are seen as too small for their surroundings ("Lilliputian")[5]. If hallucinations are constrained by a blind area of restricted size, and there is a tendency to see whole objects (rather than parts), then this would mean that hallucinated objects would have to be small simply to fit into the blind area, often too small to fit the real surroundings (e.g. a tiny hallucinated person in a real room). We can test this in our model: In the training data set, the square shape is less wide than the triangle shapes (Figure 1a). We repeated the last homeostasis experiment with the right half lesioned (9 pixels wide) instead of the top half, meaning that now only a (hallucinated) square would fit into the blind area. Indeed, we found that now, stable hallucinations are mostly squares by a large margin (Figure 4c), despite the fact that for fully blank images and for top lesioned images, squares were by far less common.[5] The network thus relied on hallucinations that fitted the blind region to restore its activity levels.

## 3.2    Cortical damage and suppression

Damage to the visual system causing complex hallucinations can also be cortical, e.g. resulting from stroke. According to [1], such stroke damage needs to be located in earlier visual areas, whereas it is higher, associative areas that are argued to be both necessary and sufficient for complex hallucinations. The interpretation is thus that the lack of bottom-up input somehow 'releases' activity in higher areas. However, in [22] transcranial magnetic stimulation was applied to early cortical areas of a CBS patient in a way thought to suppress cortical excitability. This led to a cessation of hallucinations. The authors point out that this finding contradicts the release theory. Hence, if the initial loss of vision is caused by damage to early cortical areas, complex hallucinations can form over time. If on the other hand activation in early areas is suppressed when CBS symptoms have already been developed due to for example eye disease, hallucinations cease, at least temporarily.

We reproduced these findings. Taking the first hidden layer as representing an early cortical area, we repeat the homeostasis experiment with the hidden units in that layer clamped to activations as if they were receiving no input (instead of clamping the visibles to blank images), simulating a cortical lesion. Again we found stable hallucinations to emerge in the higher layers (not shown). Then, to simulate the temporary suppression experiment in [22], we take the original model that had homeostatic regulation applied with all hidden layers intact and developed hallucinations in the process, and clamp the first hidden layer to see whether that would interfere with already established hallucinations. We found that indeed, hallucinations ceased.

We note that due to the hierarchical receptive field structure in the model, the topmost hidden layer plays a special role, its units having the largest receptive fields. We find that a DBM trained without

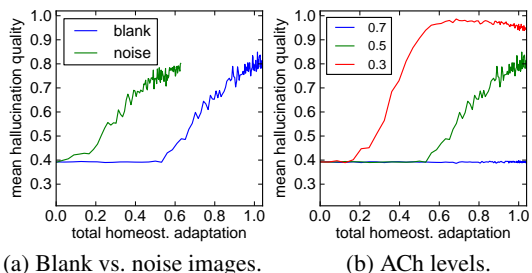

Figure 5: Comparison of hallucination emergence for blank vs. noise images *(a)* or various values of ACh balance factor $\alpha$ *(b)*. Mean qualities are plotted against the total homeostatic adaptation, defined as the absolute change of biases averaged over all units. Both noise and low $\alpha$ lower the amount of adaptation necessary to elicit hallucinations. Low $\alpha$ also increases average hallucination quality.

(a) Blank vs. noise images.  (b) ACh levels.

the topmost layer failed to learn a generative model of the toy shapes. Thus, the top-most layer or pair of layers is necessary for generating complex hallucinations, as is the case with the higher associative visual areas (although processing in the cortical hierarchy is of course much more complicated than in our model). They are also sufficient in our model in so far as homeostasis did induce hallucinations as long as the first hidden layer was clamped at the outset. However, as the last experiment has shown, when hallucinations emerge due to activity changes in the *whole* system, then interfering even with a lower layer can disrupt their formation.

We also argue that this is not merely a matter of the higher layers lacking unspecific input from lower layers. When hallucinations are formed they evoke corresponding representations in all hidden layers, even the lower ones that by themselves cannot support stable shape representations (Figure 2b). Thus, this is a result of recurrent interaction with the higher layers, likely contributing to the stability of the overall internal state. Clamping the first hidden layer prevents such recurrent stabilization. A role of recurrent interactions for hallucinations is also suggested in [23].

## 4   A novel model of acetylcholine and its role in CBS

CBS hallucinations are more likely to occur in states of drowsiness [1, 5]. This suggests a role of cholinergic and serotonergic systems, which in turn are implicated in pathologies of complex hallucinations other than CBS as well [1]. There is experimental evidence that acetylcholine (ACh) acts specifically to emphasize sensory input over internally generated one, mediating "the switching of the cortical processing mode from an intracortical to an input-processing mode" [24]. Similarly, ACh has been modeled to modulate the interaction in between bottom-up and top-down processing [25], the former delivering sensory information, the latter prior expectations.

We present a new model of ACh in the DBM framework. We take the notion that ACh influences the balance of bottom-up and top-down one step further, suggesting that in the hierarchical cortex consisting of several processing stages, ACh could mediate this balance at any stage. In the DBM model, each (intermediate) hidden layer receives input from a layer below, conveying sensory information, and from a layer above that has learned to generate or predict the former layer's activity. We thus take ACh to set the balance in between feed-forward and feed-back flow of information. To this end, we introduce a balance factor $\alpha \epsilon [0, 1]$ so that an intermediate layer $\mathbf{x}^{(k)}$ is sampled as

$$P(x_i^{(k)} = 1|\mathbf{x}^{(k-1)}, \mathbf{x}^{(k+1)}) = \sigma(\sum_j 2\alpha w_{ji}^{(k-1)} x_j^{(k-1)} + \sum_j 2(1-\alpha)w_{ij}^{(k+1)} x_j^{(k+1)}), \quad (3)$$

given states $\mathbf{x}^{(\cdot)}$ and weights $\mathbf{W}^{(\cdot)}$ above and below (biases omitted for brevity). Hence, $\alpha = 1$ equals maximal feed-forward flow of information, and $\alpha = 0.5$ recovers the normal sampling mode.

We model the effect of drowsiness on hallucinations in CBS as follows: We assume that drowsiness is being reflected as a decrease in ACh, modeled as $\alpha < 0.5$ in both intermediate hidden layers. As states of drowsiness are intermittent with periods of normal or increased vigilance (there is no pathology of these aspects in CBS per se), we assume that on average, ACh levels are still balanced. Hence, we repeat the original homeostasis experiment with bias shifts determined with $\alpha = 0.5$, but at regular intervals test the model with $\alpha = 0.3$, reflecting temporary phases of drowsiness.

Results are displayed in Figure 5b. We find that with decreased levels of ACh, not only is a much smaller homeostatic shift of excitability necessary to elicit hallucinations, but the average hallucination quality is also superior. For example, at a mean bias shift of 0.5, mean hallucination quality with $\alpha = 0.3$ is already much higher than with $\alpha = 0.5$ at maximal bias shift, whereas hallucinations

at balanced ACh levels have not even emerged yet at this point. This would thus correspond to a situation where hallucinations would only occur during drowsiness. For comparison, we also did the tests with an increased ACh level of $\alpha = 0.7$. In that case, hallucinations never emerge over the course of the homeostatic process (the end of which is determined from activities computed with $\alpha = 0.5$). In summary, we found that a temporary change in the balance of feed-forward and feedback flow of information can have a profound effect on the emergence of hallucinations, yielding a potential explanation for the role of drowsiness and ACh in CBS.

## 5   Discussion

We have reproduced a variety of findings related to CBS, and make two main predictions: First, interfering with cortical homeostatic mechanisms after the loss of vision should delay or prevent the development of hallucinations. Second, we suggest that acetylcholine could not only influence the balance of thalamic and intracortical inputs [24], but also the balance in between bottom-up and top-down at various stages of the cortical hierarchy. In CBS in particular, lack of acetylcholine at cortical sites should correlate with the emergence of hallucinations.

Neurological pathologies other than CBS have been studied before in neural networks [11]. In [18] schizophrenia is modeled with an approach akin to ours, with hallucinatory memories surfacing in a Hopfield net due to homeostatic mechanisms that compensate for input degradation. However, there the 'memories', supposedly residing in prefrontal cortex, are accounted for much more abstractly, consisting of hard-coded random patterns. In our model, these unspecified memories can be understood as learned latent representations in a hierarchical generative model of visual input. The explicit image-based representations made it possible to investigate localized degradation of visual input, and the hierarchical nature of the DBM allowed us to examine lesions and suppression within the cortex, and to model acetylcholine as mediating the feed-forward/feed-back balance of information flow. Moreover, the present work needs to be seen not just in the context of models of specifically mental dysfunction, but also in the context of models attempting to capture general principles of learning and perception in the visual cortex. Here, generative models of unsupervised learning are promising as they can naturally account for the formation of internal imagery in health and disease. We emphasize that the key aspect of a model of visual hallucinations is not that it generates images, but that it spontaneously generates rich *internal representations* of images.

We only have used toy data. As current machine learning work sees DBMs applied to more and more complex problems, more powerful demonstrations of complex hallucinations should be possible in the future. Also, other hallucinatory pathologies could be explored, such as schizophrenia. One neurological abnormality implicated in the latter is a potential disconnection of different cortical regions [26]. In the DBM, this could be modeled by decoupling different parts of the architecture, and incorporating other sensory hierarchies to account for the fact that visual hallucinations in schizophrenia tend to come with auditory hallucinations, suggesting system wide interactions.

Another interesting issue is the nature of (non-hallucinatory) mental imagery. Why is the perceptual quality of mental imagery so less salient than that of vivid hallucinations? We suggest that for mental imagery, representations are merely realized in higher areas that code for objects more abstractly, whereas for vivid hallucinations they are realized throughout the whole system [13], and hence are richer in information content. In the cortex, mechanisms such as in-built translation invariance (complex cell pooling) likely lead to some information not being represented in higher areas, something not explicitly accounted for in our model. In that context it is thus very interesting to see recent attempts [7, 27] at implementing biologically related mechanisms (such as lateral interactions) in DBM-like models that could invert this information loss when generating images: The idea is that higher layers only *seed* images in an approximate fashion, and lower areas sort out the details, by aligning edges and so forth. Then, lower areas really would be needed to realize all information entailed in rich perception, thus explaining the perceptual difference in between high level mental imagery and system wide vivid visual hallucinations.

**Acknowledgments**

We would like to thank Nicolas Heess for helpful comments, Geoff Hinton for input on the mechanism underlying the ACh model (cf. [28]), and the EPSRC, MRC and BBSRC for funding.

## Footnotes

[1]We used 1 step contrastive divergence for the greedy layer-wise training, which is an approximation to maximum likelihood gradient ascent learning, and performed no further training of the full DBM. The training set consisted of 60,000 images split into mini-batches of 100 and was iterated over through 30 epochs.

[2]To compensate for the lack of bottom-up input in this case, the weights are doubled.

[3]For computing the projections we use the original biases, not affected by homeostasis.

[4]Equation 2 is minimizing the cross entropy in between $p$ and $a$ [21].

[5]Square hallucinations were found to be least common after homeostasis over several model instances, although this bias did not exist in generations from the original models. This shows that the homeostasis induced model is not equivalent to the original one. While intriguing, we did not examine this effect further.

# References

[1] Manford, M. and Andermann, F. (1998) Complex visual hallucinations. clinical and neurobiological insights. *Brain*, **121**, 1819–1840.

[2] Collerton, D., Perry, E., and McKeith, I. (2005) Why people see things that are not there: A novel perception and attention deficit model for recurrent complex visual hallucinations. *Behavioral and Brain Sciences*, **28**, 737–757.

[3] Schultz, G. and Melzack, R. (1991) The Charles Bonnet Syndrome: 'phantom visual images'. *Perception*, **20**, 809–825, PMID: 1816537.

[4] Teunisse, R. J., Zitman, F. G., Cruysberg, J. R. M., Hoefnagels, W. H. L., and Verbeek, A. L. M. (1996) Visual hallucinations in psychologically normal people: Charles Bonnet's Syndrome. *The Lancet*, **347**, 794–797.

[5] Menon, G. J., Rahman, I., Menon, S. J., and Dutton, G. N. (2003) Complex visual hallucinations in the visually impaired: the Charles Bonnet Syndrome. *Survey of Ophthalmology*, **48**, 58–72, PMID: 12559327.

[6] Rao, R. P. and Ballard, D. H. (1999) Predictive coding in the visual cortex: a functional interpretation of some extra-classical receptive-field effects. *Nature Neuroscience*, **2**, 79–87, PMID: 10195184.

[7] Hinton, G. E. (2010) Learning to represent visual input. *Philosophical Transactions of the Royal Society B: Biological Sciences*, **365**, 177–184.

[8] Friston, K. J. (2005) Hallucinations and perceptual inference. *Behavioral and Brain Sciences*, **28**, 764–766.

[9] Corlett, P., Frith, C., and Fletcher, P. (2009) From drugs to deprivation: a Bayesian framework for understanding models of psychosis. *Psychopharmacology*, **206**, 515–530.

[10] Salakhutdinov, R. and Hinton, G. (2009) Deep Boltzmann machines. *Proceedings of the 12th International Conference on Artificial Intelligence and Statistics (AISTATS)*, vol. 5, pp. 448–455.

[11] Finkel, L. H. (2000) Neuroengineering models of brain disease. *Annual Review of Biomedical Engineering*, **2**, 577–606.

[12] Lee, H., Ekanadham, C., and Ng, A. Y. (2008) Sparse deep belief net model for visual area V2. *Advances in Neural Information Processing Systems 20*.

[13] Pollen, D. A. (1999) On the neural correlates of visual perception. *Cerebral Cortex*, **9**, 4–19.

[14] Turrigiano, G. G. and Nelson, S. B. (2000) Hebb and homeostasis in neuronal plasticity. *Current Opinion in Neurobiology*, **10**, 358–364.

[15] Houweling, A. R., Bazhenov, M., Timofeev, I., Steriade, M., and Sejnowski, T. J. (2005) Homeostatic synaptic plasticity can explain post-traumatic epileptogenesis in chronically isolated neocortex. *Cereb. Cortex*, **15**, 834–845.

[16] ffytche, D. H. and Howard, R. J. (1999) The perceptual consequences of visual loss: 'positive' pathologies of vision. *Brain*, **122**, 1247–1260.

[17] Hinton, G. E. (2007) Boltzmann machine. *Scholarpedia*, **2**, 1668.

[18] Ruppin, E., Reggia, J. A., and Horn, D. (1996) Pathogenesis of schizophrenic delusions and hallucinations: a neural model. *Schizophrenia Bulletin*, **22**, 105–123, PMID: 8685653.

[19] Hinton, G. E. and Salakhutdinov, R. R. (2006) Reducing the dimensionality of data with neural networks. *Science*, **313**, 504–507.

[20] Tsotsos, J. K., Rodriguez-Sanchez, A. J., Rothenstein, A. L., and Simine, E. (2008) The different stages of visual recognition need different attentional binding strategies. *Brain Research*, **1225**, 119–132.

[21] Nair, V. and Hinton, G. E. (2009) 3D object recognition with deep belief nets. *Advances in Neural Information Processing Systems 22*.

[22] Merabet, L. B., Kobayashi, M., Barton, J., and Pascual-Leone, A. (2003) Suppression of complex visual hallucinatory experiences by occipital transcranial magnetic stimulation: A case report. *Neurocase: The Neural Basis of Cognition*, **9**, 436.

[23] Grossberg, S. (2000) How hallucinations may arise from brain mechanisms of learning, attention, and volition. *Journal of the International Neuropsychological Society*, **6**, 583–592.

[24] Sarter, M., Hasselmo, M. E., Bruno, J. P., and Givens, B. (2005) Unraveling the attentional functions of cortical cholinergic inputs: interactions between signal-driven and cognitive modulation of signal detection. *Brain Research. Brain Research Reviews*, **48**, 98–111, PMID: 15708630.

[25] Yu, A. J. and Dayan, P. (2002) Acetylcholine in cortical inference. *Neural Networks: The Official Journal of the International Neural Network Society*, **15**, 719–730, PMID: 12371522.

[26] Ellison-Wright, I. and Bullmore, E. (2009) Meta-analysis of diffusion tensor imaging studies in schizophrenia. *Schizophrenia Research*, **108**, 3–10, PMID: 19128945.

[27] Osindero, S. and Hinton, G. (2008) Modeling image patches with a directed hierarchy of Markov random fields. *Advances in Neural Information Processing Systems*, **20**.

[28] Hinton, G. E. (2006) Unsupervised learning for perception. *NSERC Discovery Grant Proposal*, available from the author.

